# A P300 BCI for the Masses: Prior Information Enables Instant Unsupervised Spelling

**Pieter-Jan Kindermans, Hannes Verschore, David Verstraeten and Benjamin Schrauwen**
Ghent University, Electronics and Information Systems
Sint-Pietersnieuwstraat 41, 9000 Ghent, Belgium
`PieterJan.Kindermans@UGent.be`

## Abstract

The usability of Brain Computer Interfaces (BCI) based on the P300 speller is severely hindered by the need for long training times and many repetitions of the same stimulus. In this contribution we introduce a set of unsupervised hierarchical probabilistic models that tackle both problems simultaneously by incorporating prior knowledge from two sources: information from other training subjects (through transfer learning) and information about the words being spelled (through language models). We show, that due to this prior knowledge, the performance of the unsupervised models parallels and in some cases even surpasses that of supervised models, while eliminating the tedious training session.

## 1   Introduction

Brain Computer Interfaces interpret brain signals to allow direct man-machine communication [17]. In this contribution, we study the so-called P300 paradigm [6]. The user is presented with a grid of 36 characters of which alternately rows and columns light up, and focuses on the character he wishes to spell. The intensification of the focused letter can typically be detected through an event-related potential around the parietal lobe occurring 300 ms after the stimulus. By correctly detecting this so-called P300 Event Related Potential (ERP), the character intended by the user can be determined. To increase the spelling accuracy, multiple epochs are used before a character gets predicted, where a single epoch is defined as a sequence of stimuli where each row and each column is intensified once. The main difficulty in the construction of a P300 speller thus lies in the construction of a classifier for the P300 wave.

Previous work related to P300 has mainly focused on supervised training. These techniques were evaluated during several BCI Competitions [2, 3]. A popular classification method, which we will compare our proposed methods to, is Bayesian Linear Discriminant Analysis [7]. This is essentially Bayesian Linear Regression where the hyperparameters are optimized using EM [1]. It has been shown that these classifiers are among the best performing for P300 spelling [12]. A recent interesting improvement of P300 spelling is post-processing of the classifier outputs by a language model to improve spelling results [15]. Other researchers have focused on adaptive classifiers which are first trained supervisedly and then adapt to the test session while spelling [11, 13, 10]. The most flexible of these methods can be found in [11], where they are able to adapt unsupervisedly from one subject to another, however there is still need for some initial supervised training sessions.

Recent work has introduced unsupervised linear classifiers [9] that achieve accuracies comparable to state of the art supervised methods. However, these still suffer from some limitations. When the speller is used online without any prior training, it needs a warm-up period. During this warm-up period the speller output will be more or less random as the classifier is still trying to determine the underlying structure of the P300 ERP. Once the classifier has successfully learned the task, it rarely makes new mistakes. The length of this warm-up period depends on both the individual subject and

the number of epochs to spell each character. A higher number of epochs will result in fewer letters in the warm-up, but the total spelling time might increase. A second disadvantage is the fact that the classifier is randomly initialized. The remedy for this - evaluating many random initializations and selecting the best - is suboptimal and ideally one would like to choose a more intelligent initialization based on prior knowledge.

The aim of this paper is thus to reduce the warm-up period and to limit the number of initializations required to achieve acceptable performance without any subject-specific information. This will yield instant subject specific spelling, with high accuracy and a low number of epochs. To achieve this goal, we extend the graphical model of the unsupervised classifier by incorporating two types of prior knowledge: inter-subject information and language information. The key idea is that the incorporation of constraints and prior information can drastically improve a BCI's performance. The power of incorporating prior knowledge has previously been demonstrated in a BCI where finger flexion is decoded from electrocorticographic signals [18].

What we propose is a fully integrated probabilistic model, unlike previous methods which are a combination of different techniques. Furthermore, the prior work related to P300 classification possesses only a subset of the capabilities of our model.

## 2   Methods

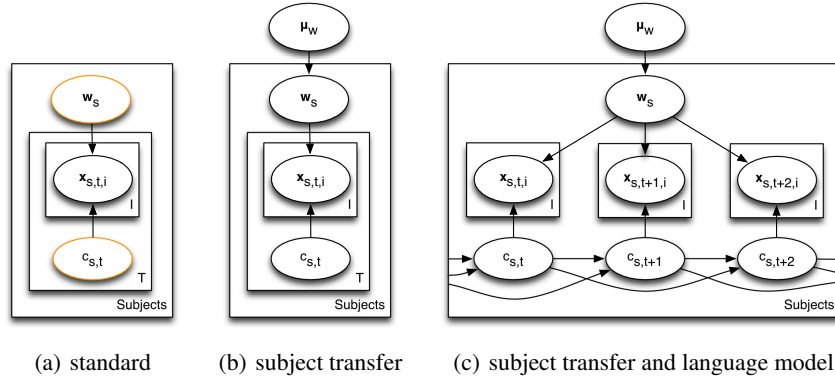

(a) standard          (b) subject transfer          (c) subject transfer and language model

Figure 1: Graphical representation of the different classifiers. On the left we show the basic unsupervised classifier [9]. In the middle we present our first contributions: the incorporation of inter subject information through a shared hyperprior. On the right the most complex model: inter subject information and a trigram language model.

### 2.1   Unsupervised P300 Speller

The basic unsupervised speller which we extend in this paper, is the unsupervised P300 classifier proposed in [9]. We will present a slightly generalized version of this model such that it does not depend on the column/row intensification structure of the default P300 application. The model is built around the following assumption: the EEG can be projected into one dimension where the projection will have a Gaussian distribution with a class dependent mean (containing P300 response versus not containing the response). From now on the distribution on the projected EEG will be used as an approximation of the distribution on the EEG itself. This makes inference and reasoning about the model simpler, but it remains an approximation. The full model, shown in Figure 1(a), is as follows:

$$p\left(\boldsymbol{w}_s\right) = \mathcal{N}\left(\boldsymbol{w}_s | \boldsymbol{0}, \alpha_s I\right), \qquad p\left(c_{s,t}\right) = \frac{1}{C},$$

$$p\left(\boldsymbol{x}_{s,t,i} | c_{s,t}, \boldsymbol{w}_s, \beta_s\right) \quad = \quad \mathcal{N}\left(\boldsymbol{x}_{s,t,i} \boldsymbol{w}_s | \boldsymbol{y}_{s,t,i}(c_{s,t}), \beta_s\right),$$

where $\boldsymbol{w}_s$ is the classifier's weight vector, $C$ is the number of symbols in the character grid, $s$ indicates the subject, $c_{s,t}$ is the $t$-th character for subject $s$. The row vector $\boldsymbol{x}_{s,t,i}$ contains the EEG recorded after intensification $i$ during spelling of $c_t$ by subject $s$, and a bias term. The EEG for a character will be denoted as $X_{s,t}$ which is a matrix whose rows are the different $\boldsymbol{x}_{s,t,i}$. Likewise, $X_s$ consists of all the features for a single subject. Both $\alpha_s$ and $\beta_s$ are values for the precision of the associated Gaussian distribution. The mean $\boldsymbol{y}_{s,t,i}(c_{s,t})$ equals 1 when during intensification $i$, character $c_{s,t}$ was highlighted, otherwise $\boldsymbol{y}_{s,t,i}(c_{s,t}) = -1$. This class dependent mean encompasses the constraint on the labeling of the individual EEG segments posed by the application: during all of the intensifications for a single character, the subject has focus on the same character. Thus during all epochs, each intensification of this character should yield the P300. Each intensification which does not include this character should not elicit a P300 response.

The probability of a character given the EEG can be computed by application of Bayes's rule: $p\left(c_{s,t}|X_{s,t}, \boldsymbol{w}_s, \beta_s\right) = \frac{p(c_{s,t})p(X_{s,t}|c_{s,t}, \boldsymbol{w}_s, \beta_s)}{\sum_{c_{s,t}} p(c_{s,t})p(X_{s,t}|c_{s,t}, \boldsymbol{w}_s, \beta_s)}$. In this model, the EEG $X_s$ contains the observed variables, the characters are the latent variables which need to be inferred and $\boldsymbol{w}_s, \beta_s, \alpha_s$ are the parameters which we want to optimize. The well known Expectation Maximization algorithm [5] can be used to optimize for $\boldsymbol{w}_s, \beta_s$ and yields following update equations:

$$\boldsymbol{w}_s = \sum_{\boldsymbol{c}_s} p\left(\boldsymbol{c}_s|X_s, \boldsymbol{w}_s^{old}, \beta_s^{old}\right) \left(X_s^T X_s + \frac{\alpha_s^{old}}{\beta_s^{old}} I\right)^{-1} X_s^T \boldsymbol{y}_s\left(\boldsymbol{c}_s\right)$$

$$\beta_s^{-1} = \left\langle \sum_{c_{s,t}} p\left(c_{s,t}|X_s, \boldsymbol{w}_s^{\text{old}}, \beta_s^{\text{old}}\right) \left(\boldsymbol{x}_{s,t,i} \boldsymbol{w}_s^{old} - \boldsymbol{y}_{s,t,i}(c_{s,t})\right)^2 \right\rangle_{t,i}$$

The update for $\boldsymbol{w}_s$ is a weighted sum ridge regression classifier trained with all possible labellings for the EEG. The weights are the probabilities that the used labels are correct given the previous weight vector $\boldsymbol{w}_s^{old}$. Let $\boldsymbol{y}(\boldsymbol{c}_s)$ be the labels which are assigned to the EEG given the character prediction $\boldsymbol{c}_s$ when the application constraints described above are taken into account. The value for $\beta_s^{-1}$ is the expected mean squared error between the projection and the target mean given the old weight vector. The hyper-parameter $\alpha_s$ can be optimized directly: $\alpha_s = \frac{D}{(\boldsymbol{w}_s^{old})^T \boldsymbol{w}_s^{old}}$, where D is the dimensionality of the weight vector. The combined optimization of $\alpha_s, \beta_s$ allows for automatic tuning of the amount of regularization but $\alpha_s$ will be bounded by $10^3$ to prevent the weight vector from collapsing onto the prior.

From the graphical representation it is clear that, without making additional assumptions about the data, there are only two ways to add additional constraints or information. First, we can incorporate prior information about the characters (the bottom of the graphical model) through language models. The second option is to incorporate prior information about the weight vector (the top of the model). We will start with the latter. Both these access points for prior knowledge are given a brighter color in Figure: 1(a).

## 2.2 Inter-subject Transfer

For the transfer learning, we drew inspiration from the work by Kemp et al. [8]. We will use hierarchical Bayesian models to share knowledge about the P300 response detection across different subjects. Our proposed model is shown in Figure 1(b) and is defined as follows:

$$p\left(\boldsymbol{\mu}_w\right) = \mathcal{N}\left(\boldsymbol{\mu}_w|0, \alpha_p I\right), \quad \alpha_p = 0, \quad p\left(\boldsymbol{w}_s|\boldsymbol{\mu}_w\right) = \mathcal{N}\left(\boldsymbol{w}_s|\boldsymbol{\mu}_w, \alpha_s I\right),$$

$$p\left(c_{s,t}\right) = \frac{1}{C}, \qquad p\left(\boldsymbol{x}_{s,t,i}|c_{s,t}, \boldsymbol{w}_s, \beta_s\right) = \mathcal{N}\left(\boldsymbol{x}_{s,t,i}\boldsymbol{w}_s|\boldsymbol{y}_{s,t,i}(c_{s,t}), \beta_s\right),$$

where we have placed a zero mean and precision Gaussian prior on the mean for the weight vector. When doing inference, we will always assume that $\boldsymbol{\mu}_w$ is given and set to its most likely value. The advantage of working with the most likely value is that there is no time penalty for transfer learning

when used in an online setting. In the case that $\boldsymbol{\mu}_w = 0$, the model reduces to the original model. On the other hand, if $\boldsymbol{\mu}_w$ takes on a nonzero value, the update equations for $\boldsymbol{w}_s, \alpha_s$ become:

$$\boldsymbol{w}_s = \sum_{\boldsymbol{c}_s} p\left(\boldsymbol{c}_s | X_s, \boldsymbol{w}_s^{old}, \beta_s^{old}\right) \left(X_s^T X_s + \frac{\alpha_s^{old}}{\beta_s^{old}} I\right)^{-1} \left(X_s^T \boldsymbol{y}_s\left(\boldsymbol{c}_s\right) + \frac{\alpha_s^{old}}{\beta_s^{old}} I \boldsymbol{\mu}_w\right),$$

$$\alpha_s = \frac{D}{\left(\boldsymbol{w}_s^{old} - \boldsymbol{\mu}_w\right)^T \left(\boldsymbol{w}_s^{old} - \boldsymbol{\mu}_w\right)}.$$

The update for $\beta_s$ remains unaltered. When we train without transfer for an initial set of subjects: $s = 1, \ldots, S$, we initialize all $\alpha_s = \alpha_p = 0$ and $\boldsymbol{\mu}_w = 0$. For this specific assignment of $\boldsymbol{\mu}_w, \alpha_p$, training is actually the same as integrating out $\boldsymbol{\mu}_w$. After the training has converged for all the subjects, we have a subject specific Maximum A Posteriori estimate: $\boldsymbol{w}_s^{new}$ and an optimized value $\alpha_s^{new}$. Using these, we can compute the posterior distribution on $\boldsymbol{\mu}_w$:

$$p\left(\boldsymbol{\mu}_w | \boldsymbol{w}_1^{new}, \ldots, \boldsymbol{w}_s^{new}\right) = \mathcal{N}\left(\boldsymbol{\mu}_w | \boldsymbol{\mu}_p^{new}, \alpha_p^{new} I\right),$$

$$\boldsymbol{\mu}_p^{new} = \frac{1}{\alpha_p^{new}} \sum_{s=1\ldots S} \alpha_s^{new} \boldsymbol{w}_s^{new}, \qquad \alpha_p^{new} = \sum_{s=1\ldots S} \alpha_s^{new}.$$

To apply transfer learning for a new subject $S+1$, we assign $\boldsymbol{\mu}_w = \boldsymbol{\mu}_p^{new}$ and keep it fixed. The new $\alpha_{S+1}$ is initialized with $\alpha_p^{new}$. The role of the optimization of $\alpha_{S+1}$ is to let the model determine whether we can build a proper model by staying close to the prior ($\alpha_{S+1}$ takes on large values) or whether we have to build a very specific model ($\alpha_{S+1}$ becomes very small).

## 2.3 Incorporation of language models

A second possibility is to incorporate language models. The only difference between working with and without a language model lies in the computation of the probability of a character given the EEG. Hence the E-step will change but the M-step will not. Please note that we have dropped the subject specific index, and we will continue to do so in this section to keep the notation uncluttered.

An n-gram language model takes the history into account: the probability of a character is defined given the $n-1$ previous characters: $p\left(c_t | c_{t-1}, \ldots, c_{t-n+1}\right)$. In this work, we limit ourselves to uni, bi and trigram language models. The graphical model of the P300 speller with subject transfer and a trigram language model is shown in Figure 1(c). For the unigram language model, which counts character frequencies, we only have to change the prior on the characters $p\left(c_t\right)$ to the probability of each character occurring.

To compute the marginal probability of a character given the EEG, which is exactly what we need in the E-step, we use the well known forward backward algorithm for HMM's [1]. For general n-grams, this algorithm computes:

$$p\left(X_1, \ldots, X_T, c_t, \ldots, c_{t-n+2}\right) = f\left(c_t, \ldots, c_{t-n+2}\right) b\left(c_t, \ldots, c_{t-n+2}\right),$$
$$f\left(c_t, \ldots, c_{t-n+2}\right) = p\left(X_1, \ldots, X_t, c_t, \ldots, c_{t-n+2}\right),$$
$$b\left(c_t, \ldots, c_{t-n+2}\right) = p\left(X_{t+1}, \ldots, X_T | c_t, \ldots, c_{t-n+2}\right).$$

The forward and backward recursion are as follows:

$$f\left(c_t, \ldots, c_{t-n+2}\right) = p\left(X_t | c_t\right) \sum_{c_{t-n+1}} p\left(c_t | c_{t-1}, \ldots, c_{t-n+1}\right) f\left(c_{t-1}, \ldots, c_{t-n+1}\right),$$

$$b\left(c_t, \ldots, c_{t-n+2}\right) = \sum_{c_{t+1}} p\left(X_{t+1} | c_{t+1}\right) p\left(c_{t+1} | c_t, \ldots, c_{t-n+2}\right) b\left(c_{t+1}, \ldots, c_{t-n+3}\right).$$

The initialization of the forward and backward recursion is analogous to the initialization for the default HMM [1]. The probability of a character can be computed as follows:

$$p(c_t | \boldsymbol{X}) = \sum_{c_{t-1}, \ldots, c_{t-n+2}} \frac{p\left(X_1, \ldots, X_T, c_t, \ldots, c_{t-n+2}\right)}{p\left(X\right)}$$

$$= \sum_{c_{t-1}, \ldots, c_{t-n+2}} \frac{f\left(c_t, \ldots, c_{t-n+2}\right) b\left(c_t, \ldots, c_{t-n+2}\right)}{\sum_{c_t, \ldots, c_{t-n+2}} f\left(c_t, \ldots, c_{t-n+2}\right) b\left(c_t, \ldots, c_{t-n+2}\right)}.$$

This can then be plugged directly into the EM-update equations from Section 2.1. Note that when we cache the forward pass from previous character predictions, only a single step of both the forward and backward pass has to be executed to spell a new character.

# 3 Experiments and Discussion

## 3.1 The Akimpech Dataset

We performed our experiments on the public Akimpech P300 database [19]. This dataset covers 22 subjects[1] who spelled Spanish words. The data was recorded with a 16 channel g.tec gUSBamp EEG amplifier at 256 Hz but only 10 channels are available in the dataset. The recording was performed with the BCI2000 P300 speller software [14] with the following settings: a 2 second pause before and after each character, 62.5ms between the intensifications, these intensifications lasted 125ms each and the spelling matrix contained the characters $[a - z1 - 9\_]$. The dataset comprises both a train and a test set. The train set contains 16 characters with 15 epochs per character. This train set will not be used by the unsupervised classifiers but only by the supervised classifier which we will later use for comparison. The number of characters in the test is subject dependent and ranges from 17 to 29 with an average of 22.18. This limited number of characters per sequence is very challenging for our unsupervised classifier, since the spelling has to be as correct as possible right from the start, in order to obtain high average accuracies.

As the pre-processing in [9] has been shown to lead to good spelling performance, we adhere to their approach. The EEG is preprocessed one character at a time; as a consequence this approach is valid in real online experiments[2]. Pre-processing begins by applying a Common Average Reference filter, followed by a bandpass filter (0.5 Hz - 15 Hz). Afterwards, each EEG channel is normalized such that it has zero mean and unit variance. The final step is sub-sampling the EEG by a factor 6 and retaining 10 samples which are centered at 300ms after stimulus presentation.

## 3.2 Training the Language models and Spelling Real Text

The Akimpech dataset was recorded using a limited number of Spanish words with an unrepresentative subset of characters. It is therefore not an accurate representation of how a realistic speller would be used. To alleviate this, we constructed a dataset which contains words that would be spelled in a realistic context. This is done by re-synthesizing a dataset using the EEG from the Akimpech dataset and sentences from the English Wikipedia dataset from Sutskever et al. [16].

In a P300 speller, a look-up table assigns a specific character to each position in the on-screen matrix. The actual task is to determine the position that, when intensified, evokes the P300 response. To spell a symbol, we predict the desired position, then we look up the symbol assigned to it. Thus, in a standard P300 setup the desired text can be modified by altering the look-up table. Furthermore, this will not influence the performance as long as the desired symbol is assigned to a single position. This approach remains valid when language models are integrated into the classifier, because neither the EEG nor the intensification structure is modified.

The Wikipedia dataset was transformed to lowercase and we used the first $5 \cdot 10^8$ characters in the dataset to select the 36 most frequently occurring characters excluding numeric symbols. We argue that using a subset of numbers is of no use and since we add the space as a symbol, we have to drop at least one numeric symbol. As such, it makes sense to replace all the numeric characters with other symbols. The selected characters are the following: $[a - z : \%()' - ".,\_]$, where the underscore signifies whitespace. This set of characters is then used to train unigram, bigram and trigram letter models. These language models were trained on the first $5 \cdot 10^8$ characters and we applied Witten-Bell smoothing [4], which assigns small but non-zero probabilities to n-grams not encountered in the train set.

The remaining part of the Wikipedia dataset was used to generate target texts for classifier evaluation. This part was not used to train the language models. First we dropped the non-selected characters. Then for each subject, we sampled new texts, with the same length as the originally spelled text,

from the dataset. Additionally, we modified the contents of the character grid such that it contains the 36 selected symbols. The look-up table for the individual spelling actions was changed such that the correct solution is the newly sampled text. This is implemented by taking the base look-up table, with $[a - z : \%()' - "., \_]$, with $a$ in the top left and $\_$ in the bottom right corner of the screen, and cyclically shifting it.

## 3.3 Experimental setup

We tested 12 different classifiers, where we use the following code to name the classifiers. The first letter indicates how the classifier is initialized, either randomly (R) or using subject Transfer (T). The second letter indicates whether the classifiers adapts unsupervisedly during the spelling session (A) or is static (S). We compared the standard unsupervised (and adaptive) algorithm (RA) which is randomly initialized, our proposed transfer learning approach without online adaptation (TS) and the transfer learning approach with adaptation (TA). These three different setups were tested without a language model, and with a uni-, bi- and trigram language model. We will indicate the language model by appending the subscript '$-$' for the classifier without language model, '$uni$' for unigram, etcetera. For example: $TA_{tri}$ is the unsupervised classifier which used transfer learning, learns on the fly and includes a trigram language model. The classifier $RA_-$ is the baseline which we want to improve on.

The influence of performance fluctuations caused by the initialization or desired text is minimized as follows. We executed 20 experiments per subject, where in each experiment all the classifiers are evaluated. The desired text and classifier initializations are experiment specific. This means that for each subject we have 20 desired texts, 20 random initializations and 20 subject transfer initializations. Each classifier was evaluated on all of these texts, where for each text we always used the same initialization. Additionally, we repeated the experiments with 3, 4, 5, 10 and 15 epochs per character.

The randomly initialized adaptive procedures work as in [9]. In short, the classifier first receives the EEG for the next character. The EEG is added to the unsupervised trainset and 3 EM iterations are executed[3]. Next, the desired symbol is predicted with the updated classifier.

In the case of transfer learning the initializations are computed as discussed in section 2.2. The initial classifiers used in the transfer learning process itself were trained unsupervisedly and offline without a language model. For each subject, we drew 5 samples for $w$ and trained 2 classifiers per draw: one with $w$ and one with $-w$ such that at least one is above chance level for the binary P300 detection. From the resulting 10 classifiers we selected the one which has the highest log likelihood, to be used in transfer learning. Finally, the current test subject is omitted when computing the transfer learning parameters. In short: the transfer learning parameters are computed without seeing labeled data and more importantly without seeing any data from the current subject.

We conclude this section by discussing the time complexity of the methods. The use of transfer learning does not increase the time needed to predict a character. However the time needed per EM iteration scales linearly with the number of characters in the trainset. The addition of n-gram language models scales the time per E-step with (number of characters in grid)$^{n-1}$. Therefore character prediction can become very time consuming. As this is a major issue in this real-time application, we will also discuss the setting where the classifier is first used to spell the next character and the EM updates are executed during the intensifications for the following symbol. As mentioned in Section 2.3, only a single step in the forward and backward pass is needed to spell the next character. Thus we can state that this approach yields instantaneous spelling. This classifier will be named $TA^*$.

## 3.4 Results

We will start the discussion of the results with the baseline method $RA_-$ followed by the evaluation of our contributions. An overview of the averaged results of all online experiments is available in Figure 2. In Figure 3 we show the performance on the test set after the classifiers have processed the test set and adapted to it, if possible. When we retest the adapted classifier we will denote this by appending '-R' to its name.

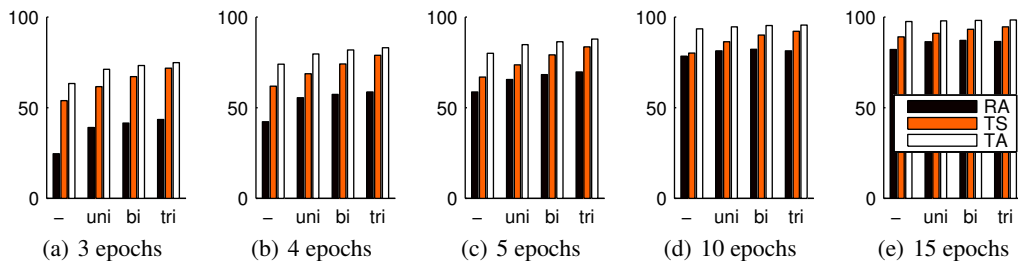

(a) 3 epochs  (b) 4 epochs  (c) 5 epochs  (d) 10 epochs  (e) 15 epochs

Figure 2: Overview of all spelling results from online experiments. Increasing the number of epochs or adding complex language models improves accuracy. Furthermore, transfer learning without adaptation (TS) outperforms learning from scratch (RA). Adding adaptation to the transfer learning improves the results even further (TA).

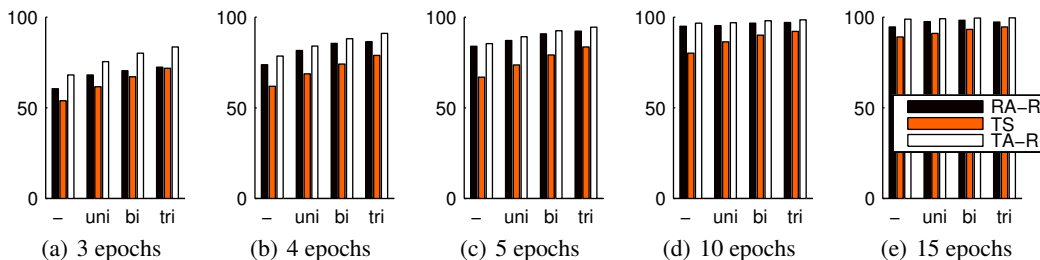

(a) 3 epochs  (b) 4 epochs  (c) 5 epochs  (d) 10 epochs  (e) 15 epochs

Figure 3: Spelling accuracy when the test set is processed online and the classifiers are re-evaluated afterwards. In Figure 2 we saw that the TS approach outperformed the RA range of classifiers. Here we see that TA-R and RA-R outperform TS even with few epochs. It is also clear that the adaptive classifiers are able to correct mistakes they made initially.

Application of the baseline method $RA_-$ and averaging the results over the different subjects results in an online spelling accuracy starting at 24.6% for 3 epochs and up to 82.1% for 15 epochs. The result with 15 epochs is usable in practice and predicts only 4 characters incorrectly. However, the spelling time is about half a minute per character. Retesting the classifiers obtained after the online experiment gives the following results: when 3 epochs are used the final classifier is able to spell 60.5% correctly, for 15 epochs this becomes 94.6%. This corroborates the findings from the original paper [9] that the classifiers need the warm-up period before they start to produce correct predictions.

By evaluating the addition of a language model, $RA_{uni,bi,tri}$, we see an improvement of the online results. The longer the time dependency in the language model, the bigger the improvement. As more repetitions are used per character, the performance gain of the language models diminishes. For 3 repetitions, a tri-gram model produces an online spelling accuracy of 43.5% compared to 24.6% without a language model. The results for 15 repetitions show that on average 3 characters are predicted incorrectly when a trigram is used. Analysis of the re-evaluation of the classifiers after online processing shows a smaller improvement to the results, indicating that the language model mainly helps to reduce the warm up period.

Next we consider the influence of transfer learning. We begin by evaluating the TS classifiers, which do not use unsupervised adaptation. Overall, TS classifiers outperform the RA range, even when the latter uses a trigram model. However, the post-test reevaluation shows that the RA methods are able to outperform TS. In essence: given enough data, the adaptive method has the ability to learn a better model than the transfer learning approach. Addition of the language models to the TS classifier shows a secondary improvement, as is to be expected.

This brings us to the full model TA: adaptive unsupervised training which is initialized with transfer learning and optionally makes use of language models. Figures 2 and 3 indeed confirm that these

Table 1: Comparison between different classifiers. The BLDA classifiers are subject-specific and supervisedly trained. $\text{BLDA}_{tri}^-$ was trained using 3 epochs. The basic $\text{RA}_-$ and the full model $\text{TA}_{tri}$ are included. Furthermore we give results for an adapted version $\text{TA}_{tri}^*$, which spells the character before the EM updates and for $\text{TA-R}_{tri}$, which is the re-evaluation of $\text{TA}_{tri}$ after processing the test set.

| Epochs | $\text{RA}_-$ | $\text{TA}_{tri}^*$ | $\text{TA}_{tri}$ | $\text{TA-R}_{tri}$ | BLDA | $\text{BLDA}_{tri}$ | $\text{BLDA}_{tri}^-$ |
|---|---|---|---|---|---|---|---|
| 3 | 24.6 | 73.8 | 74.8 | 83.5 | 74.5 | 89.4 | 78.9 |
| 4 | 42.2 | 82.1 | 83.0 | 91.0 | 82.2 | 93.0 | 83.8 |
| 5 | 58.6 | 87.0 | 87.8 | 94.4 | 84.9 | 94.6 | 86.5 |
| 10 | 78.4 | 95.0 | 95.5 | 98.5 | 93.0 | 97.4 | 92.5 |
| 15 | 82.1 | 97.9 | 98.4 | 99.5 | 96.7 | 98.1 | 94.3 |

models produce the best results both in the online test and in the re-evaluation afterwards, when we consider unsupervised spelling. Also, the trigram classifier produces the best results, which is not surprising given the incorporation of important prior language knowledge into the model.

Next, we give an overview of spelling accuracies in Table 1, where we compare the basic unsupervised method $\text{RA}_-$ to the full model $\text{TA}_{tri}$. With nearly three times as accurate spelling for 3 epochs (74.8% compared to 24.6%) and near perfect spelling for 15 epochs, we can conclude that the full model is capable of instant spelling for a novel subject. The application of $\text{TA}_{tri}^*$ results in a minute performance drop, but as this classifier spells the character before performing the EM iterations, it allows for real-time spelling when the EEG is received and is therefore of more use in an online setting.

To conclude, we compare the unsupervised methods with BLDA, which is the supervised counterpart of the $\text{RA}_-$ classifier. The BLDA classifiers in this table are supervisedly trained using 15 epochs per character on 16 characters. This is slightly over 10 minutes of training before one can start spelling. The $\text{BLDA}_{tri}^-$ classifier used a limited training set with only 3 epochs per character or almost three minutes of training. When the limited training set is used, we see that our proposed method produces results which are competitive for 3-5 epochs and better for 10 and 15. The $\text{BLDA}_{tri}$ model outperforms our method when we consider a low number of repetitions per character but not for 10 or 15 epochs. From 4 epochs onwards we can see that the re-evaluated classifier after online learning ($\text{TA-R}_{tri}$) is able to learn models which are as good as supervisedly trained models. Finally we would like to point out that even for just 3 epochs per character, our proposed method spelled less characters wrongly (about 6 on average) than the number of characters used during the supervised training (16 for each subject).

## 4 Conclusion

In this work we set out to build a P300 based BCI which is able to produce accurate spelling for a novel subject without any form of training session. This is made possible by incorporating both inter-subject information and language models directly into an unsupervised classifier. This yields a coherent probabilistic model which quickly adapts to unseen subjects, by exploiting several forms of prior information. This contrasts with all supervised methods which need time consuming training session. There are only a few other unsupervised approaches for P300 spelling, but they need a warm-up period during which the speller is unreliable or they need labeled data to initialize the adaptive spellers. We compared our method to the original unsupervised speller proposed in [9] and have shown that unlike theirs, our approach works instantly. Furthermore, our final experiments demonstrated that the proposed method can compete with state of the art subject-specific and supervisedly trained classifiers [7], even when incorporating a language model.

**Acknowledgments**

This work was partially funded by the Ghent University Special Research Fund under the BOF-GOA project Home-MATE.

## Footnotes

[1]There are more subjects listed on the website but some files are corrupt.

[2]This claim was empirically verified, we omit the discussion of these experiments due to page constraints.

[3]This is a trade-off between classifier update time and performance.

# References

[1] C. M. Bishop. *Pattern Recognition and Machine Learning (Information Science and Statistics)*. Springer, 1 edition, 2007.

[2] B. Blankertz, K.-R. Muller, G. Curio, T.M. Vaughan, G. Schalk, J.R. Wolpaw, A. Schlogl, C. Neuper, G. Pfurtscheller, T. Hinterberger, M. Schroder, and N. Birbaumer. The BCI competition 2003: progress and perspectives in detection and discrimination of EEG single trials. *IEEE Trans. on Biomedical Engineering*, 51(6):1044 –1051, June 2004.

[3] B. Blankertz, K.-R. Muller, D.J. Krusienski, G. Schalk, J.R. Wolpaw, A. Schlogl, G. Pfurtscheller, Jd.R. Millan, M. Schroder, and N. Birbaumer. The BCI competition III: validating alternative approaches to actual BCI problems. *IEEE Trans. on Neural Systems and Rehabilitation Engineering*, 14(2):153 –159, June 2006.

[4] S.F. Chen and J. Goodman. An empirical study of smoothing techniques for language modeling. *Computer Speech & Language*, 13(4):359–393, 1999.

[5] A. P. Dempster, N. M. Laird, and D. B. Rubin. Maximum likelihood from incomplete data via the EM algorithm. *Journal of the Royal Statistical Society. Series B (Methodological)*, 39(1):pp. 1–38, 1977.

[6] L.A. Farwell and E. Donchin. Talking off the top of your head: toward a mental prosthesis utilizing event-related brain potentials. *Electroencephalography and Clinical Neurophysiology*, 70(6):510 – 523, 1988.

[7] U. Hoffmann, J.-M. Vesin, T. Ebrahimi, and K. Diserens. An efficient P300-based brain–computer interface for disabled subjects. *Journal of Neuroscience Methods*, 167(1):115 – 125, 2008.

[8] C. Kemp, A. Perfors, and J.B. Tenenbaum. Learning overhypotheses with hierarchical bayesian models. *Developmental science*, 10(3):307–321, 2007.

[9] P.-J. Kindermans, D. Verstraeten, and B. Schrauwen. A bayesian model for exploiting application constraints to enable unsupervised training of a P300-based BCI. *PLoS ONE*, 7(4):e33758, 04 2012.

[10] Y. Li, C. Guan, H. Li, and Z. Chin. A self-training semi-supervised SVM algorithm and its application in an EEG-based brain computer interface speller system. *Pattern Recognition Letters*, 29(9):1285 – 1294, 2008.

[11] S. Lu, C. Guan, and H. Zhang. Unsupervised brain computer interface based on intersubject information and online adaptation. *IEEE Trans. on Neural Systems and Rehabilitation Engineering*, 17(2):135 –145, 2009.

[12] N.V. Manyakov, N. Chumerin, A. Combaz, and M.M. Van Hulle. Comparison of linear classification methods for P300 brain-computer interface on disabled subjects. *BIOSIGNALS, Rome, Italy*, pages 328–334, 2011.

[13] R.C. Panicker, S. Puthusserypady, and Ying S. Adaptation in P300 brain-computer interfaces: A two-classifier cotraining approach. *IEEE Trans. on Biomedical Engineering*, 57(12):2927 –2935, December 2010.

[14] G. Schalk, D. J. Mcfarland, T. Hinterberger, N. Birbaumer, and J. R. Wolpaw. BCI2000: A general-purpose brain-computer interface (BCI) system. *IEEE Trans. on Biomedical Engineering*, 51:2004, 2004.

[15] W. Speier, C. Arnold, J. Lu, R. K. Taira, and N. Pouratian. Natural language processing with dynamic classification improves P300 speller accuracy and bit rate. *Journal of Neural Engineering*, 9(1):016004, 2012.

[16] I. Sutskever, J. Martens, and G. Hinton. Generating text with recurrent neural networks. In *International Conference on Machine Learning (ICML)*, 2011.

[17] J. J. Vidal. Toward direct brain-computer communication. *Annual Review of Biophysics and Bioengineering*, 2(1):157–180, 1973.

[18] Z. Wang, G. Schalk, and Q. Ji. Anatomically constrained decoding of finger flexion from electrocorticographic signals. In J. Shawe-Taylor, R.S. Zemel, P. Bartlett, F.C.N. Pereira, and K.Q. Weinberger, editors, *Advances in Neural Information Processing Systems 24*, pages 2070–2078. 2011.

[19] O. Yanez-Suarez, L. Bougrain, C. Saavedra, E. Bojorges, and G. Gentiletti. P300-speller public-domain database, 5 2012.

